# Tractable Variational Structures for Approximating Graphical Models

David Barber      Wim Wiegerinck

{davidb,wimw}@mbfys.kun.nl

RWCP* Theoretical Foundation SNN† University of Nijmegen

6525 EZ Nijmegen, The Netherlands.

## Abstract

Graphical models provide a broad probabilistic framework with applications in speech recognition (Hidden Markov Models), medical diagnosis (Belief networks) and artificial intelligence (Boltzmann Machines). However, the computing time is typically exponential in the number of nodes in the graph. Within the variational framework for approximating these models, we present two classes of distributions, decimatable Boltzmann Machines and Tractable Belief Networks that go beyond the standard factorized approach. We give generalised mean-field equations for both these directed and undirected approximations. Simulation results on a small benchmark problem suggest using these richer approximations compares favorably against others previously reported in the literature.

## 1 Introduction

Graphical models provide a powerful framework for probabilistic inference[1] but suffer intractability when applied to large scale problems. Recently, variational approximations have been popular [2, 3, 4, 5], and have the advantage of providing rigorous bounds on quantities of interest, such as the data likelihood, in contrast to other approximate procedures such as Monte Carlo methods[1]. One of the original models in the neural networks community, the Boltzmann machine (BM), belongs to the class of undirected graphical models. The lack of a suitable algorithm has hindered its application to larger problems. The deterministic BM algorithm[6], a variational procedure using a factorized approximating distribution, speeds up the learning of BMs, although the simplicity of this approximation can lead to undesirable effects[7]. Factorized approximations have also been successfully applied to sigmoid belief networks[4]. One approach to producing a more accurate approximation is to go beyond the class of factorized approximating models by using, for example, mixtures of factorized models. However, it may be that very many mixture components are needed to obtain a significant improvement beyond using the factorized approximation[5]. In this paper, after describing the variational learn-

ing framework, we introduce two further classes of non-factorized approximations, one undirected (decimatable BMs in section (3)) and the other, directed (Tractable Belief Networks in section (4)). To demonstrate the potential benefits of these methods, we include results on a toy benchmark problem in section (5) and discuss their relation to other methods in section (6).

## 2   Variational Learning

We assume the existence of a graphical model $P$ with known qualitative structure but for which the quantitative parameters of the structure remain to be learned from data. Given that the variables can be considered as either visible ($V$) or hidden ($H$), one approach to learning is to carry out maximum likelihood on the visible variables for each example in the dataset. Considering the KL divergence between the true distribution $P(H|V)$ and a distribution $Q(H)$,

$$KL(Q(H), P(H|V)) = \sum_H Q(H) \ln \frac{Q(H)}{P(H|V)} \geq 0$$

and using $P(H|V) = P(H,V)/P(V)$ gives the bound

$$\ln P(V) \geq -\sum_H Q(H) \ln Q(H) + \sum_H Q(H) \ln P(H,V) \tag{1}$$

Betraying the connection to statistical physics, the first term is termed the "entropy" and the second the "energy". One typically chooses a variational distribution $Q$ so that the entropic term is "tractable". We assume that the energy $E(Q)$ is similarly computable, perhaps with recourse to some extra variational bound (as in section (5)). By tractable, we mean that all necessary marginals and desired quantities are computationally feasible, regardless of the issue of the scaling of the computational effort with the graph size. Learning consists of two iterating steps: first optimize the bound (1) with respect to the parameters of $Q$, and then with respect to the parameters of $P(H,V)$. We concentrate here on the first step. For clarity, we present our approach for the case of binary variables $s_i \in \{0,1\}, i = 1..N$. We now consider two classes of approximating distributions $Q$.

## 3   Undirected Q: Decimatable Boltzmann Machines

Boltzmann machines describe probability distributions parameterized by a symmetric weight matrix $J$

$$Q(s) = \frac{1}{Z} \exp \phi, \qquad \phi \equiv \sum_{ij} J_{ij} s_i s_j = s \cdot J s \tag{2}$$

where the normalization constant, or "partition function" is $Z = \sum_s \exp \phi$. For convenience we term the diagonals of $J$ the "biases", $h_i = J_{ii}$. Since $\ln Z(J,h)$ is a generating function for the first and second order statistics of the variables $s$, the entropy is tractable provided that $Z$ is tractable. For general connection structures, $J$, computing $Z$ is intractable as it involves a sum over $2^N$ states; however, not all Boltzmann machines are intractable. A class of tractable structures is described by a set of so-called decimation rules in which nodes from the graph can be removed one by one, fig(1). Provided that appropriate local changes are made to the BM parameters, the partition function of the reduced graph remains unaltered (see eg [2]). For example, node $c$ in fig(1) can be removed, provided that the weight matrix $J$ and bias $h$ are transformed, $J \to J'$, $h \to h'$, with $J'_{ac} = J'_{bc} = h'_c = 0$ and

$$J'_{ab} = J_{ab} + \frac{1}{2} \ln \frac{(1+e^{h_c})(1+e^{h_c+2(J_{ac}+J_{bc})})}{(1+e^{h_c+2J_{ac}})(1+e^{h_c+2J_{bc}})}, \qquad h'_{a/b} = h_{a/b} + \ln \frac{1+e^{h_c+2J_{a/b,c}}}{1+e^{h_c}} \tag{3}$$

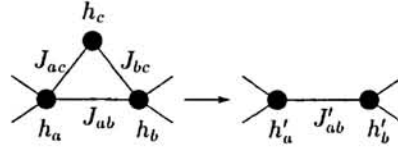

Figure 1: A decimation rule for BMs. We can remove the upper node on the left so that the partition function of the reduced graph is the same. This requires a simple change in the parameters $J, h$ coupling the two nodes on the right (see text).

By repeatedly applying such rules, $Z$ is calculable in time linear in $N$.

## 3.1 Fixed point (Mean Field) Equations

Using (2) in (1), the bound we wish to optimize with respect to the parameters $\theta = (J, h)$ of $Q$ has the form ($\langle \ldots \rangle$ denotes averages with respect to $Q$)

$$B(\theta) = -\langle \phi \rangle + \ln Z + E(\theta) \qquad (4)$$

where $E(\theta)$ is the energy. Differentiating (4) with respect to $J_{ij} (i \neq j)$ gives

$$\frac{\partial B}{\partial J_{ij}} = -\sum_{kl} F_{ij,kl} J_{kl} + \frac{\partial E}{\partial J_{ij}} \qquad (5)$$

where $F_{ij,kl} = \langle s_i s_j s_k s_l \rangle - \langle s_i s_j \rangle \langle s_k s_l \rangle$ is the Fisher information matrix. A similar expression holds for the bias parameters, $h$, so that we can form a linear fixed point equation in the total parameter set $\theta$ where the derivatives of the bound vanish. This suggests the iterative solution, $\theta^{new} = F^{-1} \nabla_\theta f$ where the right hand side is evaluated at the current parameter values, $\theta^{old}$.

## 4   Directed Q: Tractable Belief Networks

Belief networks are products of conditional probability distributions,

$$Q(H) = \prod_{i \in H} Q(H_i | \pi_i) \qquad (6)$$

in which $\pi_i$ denotes the parents of node $i$ (see for example, [1]). The efficiency of computation depends on the underlying graphical structure of the model and is exponential in the maximal clique size (of the moralized triangulated graph [1]). We now assume that our model class consists of belief networks with a fixed, tractable graphical structure. The entropy can then be computed efficiently since it decouples into a sum of averaged entropies per site $i$ ($Q(\pi_i) \equiv 1$ if $\pi_i = \phi$),

$$\sum_H Q(H) \ln Q(H) = \sum_{i \in H} \sum_{\pi_i} Q(\pi_i) \sum_{H_i} Q(H_i | \pi_i) \ln Q(H_i | \pi_i) \qquad (7)$$

Note that the conditional entropy at each site $i$ is trivial to compute since the values required can be read off directly from the definition of $Q$ (6). By assumption, the marginals $Q(\pi_i)$ are tractable, and can be found by standard methods, for example using the Junction Tree Algorithm[1].

To optimize the bound (1), we parameterize $Q$ via its conditional probabilities, $q_i(\pi_i) \equiv Q(H_i = 1 | \pi_i)$. The remaining probability $Q(H_i = 0 | \pi_i)$ follows from

normalization. We therefore have a set $\{q_i(\pi_i)|\pi_i = (0\ldots0),\ldots,(1\ldots1)\}$ of variational parameters for each node in the graph. Setting the gradient of the bound with respect to the $q_i(\pi_i)$'s equal to zero yields the equations

$$q_i(\pi_i) = \sigma\left(\frac{(\nabla_{i\pi_i} E(Q)) + L_{i\pi_i}}{Q(\pi_i)}\right) \tag{8}$$

with

$$L_{i\pi_i} = -\sum_j \sum_{\pi_j} [\nabla_{i\pi_i} Q(\pi_j)] \sum_{H_j} Q(H_j|\pi_j) \ln Q(H_j|\pi_j) \tag{9}$$

where $\sigma(z) = 1/(1 + e^{-z})$. The gradient $\nabla_{i\pi_i}$ is with respect to $q_i(\pi_i)$. The explicit evaluation of the gradients can be performed efficiently, since all that need to be differentiated are at most scalar functions of quantities that depend again only linearly on the parameters $q_i(\pi_i)$. To optimize the bound, we iterate (8) till convergence, analogous to using factorized models[4]. However, the more powerful class of approximating distributions described by belief networks should enable a much tighter bound on the likelihood of the visible units.

## 5  Application to Sigmoid Belief Networks

We now describe an application of these non-factorized approximations to a particular class of directed graphical models, sigmoid belief networks[8] for which the conditional distributions have the form

$$P(s_i = 1|\pi_i) = \sigma\left(\sum_j W_{ij} s_j + k_i\right) \tag{10}$$

$W_{ij} = 0$ if $j \notin \pi_i$. The joint distribution then has the form

$$P(H, V) = \prod_i \exp\left[z_i s_i - \ln(1 + e^{z_i})\right] \tag{11}$$

where $z_i = \sum_j W_{ij} s_j + k_i$. In (11) it is to be understood that the visible units are set to their observed values. In the lower bound (1), unfortunately, the average of $\ln P(H, V)$ is not tractable, since $\langle \ln[1 + e^z] \rangle$ does not decouple into a polynomial number of single site averages. Following [4] we use therefore the bound

$$\langle \ln[1 + e^z] \rangle \le \xi \langle z \rangle + \ln\left\langle e^{-\xi z} + e^{(1-\xi)z} \right\rangle \tag{12}$$

where $\xi$ is a variational parameter in $[0, 1]$. We can then define the energy function

$$E(Q, \xi) = \sum_{ij} W_{ij} \langle s_i s_j \rangle + \sum_i \tilde{k}_i \langle s_i \rangle - \sum_i k_i \xi_i - \sum_i \ln\left\langle e^{-\xi_i z_i} + e^{(1-\xi_i)z_i} \right\rangle \tag{13}$$

where $\tilde{k}_i = k_i - \sum_j \xi_j W_{ji}$. Expect for the final term, the energy is a function of first or second order statistics of the variables. For using a BM as the variational distribution, the final terms of (13) $\langle e^{-\xi_i z_i} \rangle = \sum_H e^{\phi - \xi_i z_i}/Z$ are simply the ratio of two partition functions, with the one in the numerator having a shifted bias. This is therefore tractable, provided that we use a tractable BM $Q$.

Similarly, if we are using a Belief Network as the variational distribution, all but the last term in (13) is trivially tractable, provided that $Q$ is tractable. We write the terms $\langle e^{-\xi_i z_i} \rangle = e^{-\xi_i h_i} \sum_H R(H)$, where $R(H) = \prod_j R(H_j|\pi_j)$ and $R(H_j|\pi_j) \equiv$

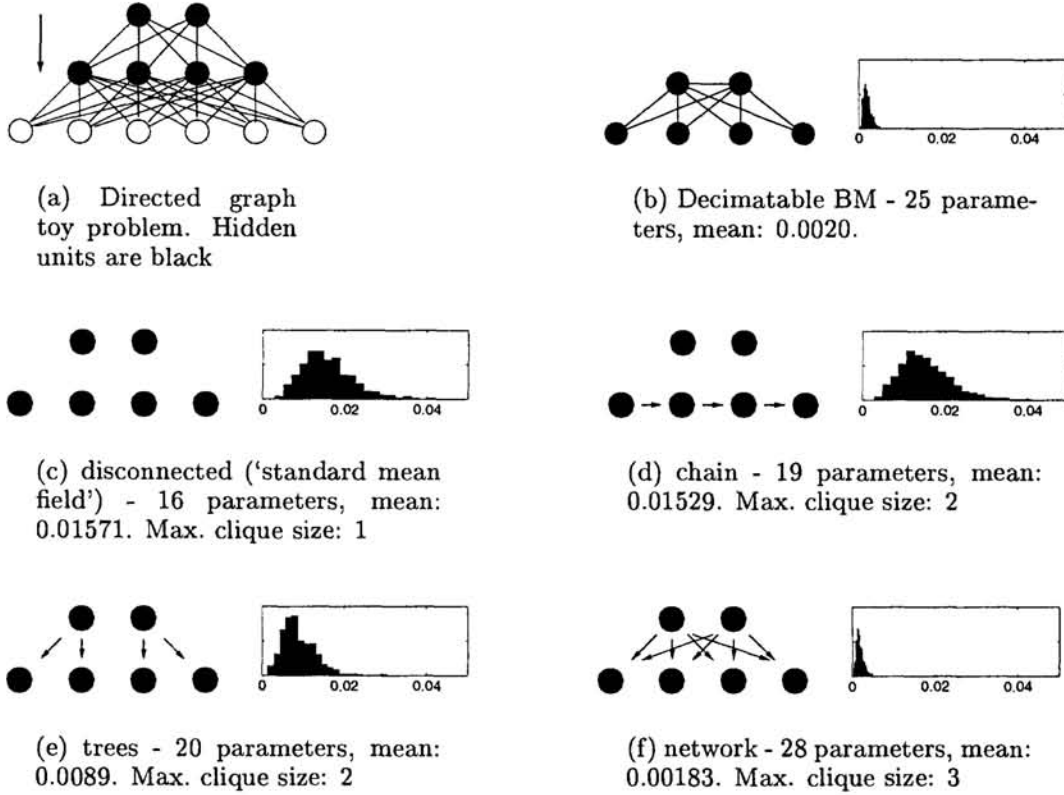

(a) Directed graph toy problem. Hidden units are black

(b) Decimatable BM - 25 parameters, mean: 0.0020.

(c) disconnected ('standard mean field') - 16 parameters, mean: 0.01571. Max. clique size: 1

(d) chain - 19 parameters, mean: 0.01529. Max. clique size: 2

(e) trees - 20 parameters, mean: 0.0089. Max. clique size: 2

(f) network - 28 parameters, mean: 0.00183. Max. clique size: 3

Figure 2: (a) Sigmoid Belief Network for which we approximate $\ln P(V)$. (b): BM approximation. (c,d,e,f): Structures of the directed approximations on $H$. For each structure, histograms of the relative error between the true log likelihood and the lower bound is plotted. The horizontal scale has been fixed to $[0,0.05]$ in all plots. The maximum clique size refers to the complexity of computation for each approximation, which is exponential in this quantity. The number of parameters includes the vector $\xi$.

$Q(H_j|\pi_j) \exp(-\xi_i J_{ij} H_j)$. $R$ and $Q$ have the same graphical structure and we can therefore use message propagation techniques again to compute $\langle e^{-\xi_i z_i} \rangle$.

To test our methods numerically, we generated 500 networks with parameters $\{W_{ij}, k_j\}$ drawn randomly from the uniform distribution over $[-1, 1]$. The lower bounds $\mathcal{F}_V$ for several approximating structures are compared with the true log likelihood, using the relative error $\mathcal{E} = \mathcal{F}_V / \ln P(V) - 1$, fig. 2. These show that considerable improvements can be obtained when non-factorized variational distributions are used. Note that a 5 component mixture model ($\approx 80$ variational parameters) yields $\mathcal{E} = 0.01139$ on this problem [5][1]. These results suggest therefore that exploiting knowledge of the graphical structure of the model is useful. For instance, the chain (fig. 2(b)) with no graphical overlap with the original graph shows hardly any improvement over the standard mean field approximation. On the other hand, the tree model (fig. 2(c)), which has about the same number of parameters, but a larger overlap with the original graph, does improve considerably over the mean field approximation (and even over the 5 component mixture model). By increasing the overlap, as in fig. 2(d), the improvement gained is even greater.

## 6   Discussion

In this section, we briefly explain the relationship of the introduced methods to other, "non-factorized" methods in the literature, namely node-elimination[9] and substructure variation[10].

### 6.1   Graph Partitioning and Node Elimination

A further class of approximating distributions $Q$ that could be considered are those in which the nodes can be partitioned into clusters, with independencies between the clusters. For expositional clarity, consider two partitions, $s = (s_1, s_2)$, and define $Q$ to be factorized over these partitions[2], $Q = Q_1(s_1)Q_2(s_2)$. Using this $Q$ in (1), we obtain (with obvious notational simplifications)

$$\ln P(V) \geq - \langle \ln Q_1 \rangle_1 - \langle \ln Q_2 \rangle_2 + \langle \ln P \rangle_{1,2} \qquad (14)$$

A functional derivative with respect to $Q_1$ and $Q_2$ gives the optimal forms:

$$Q_1 = \exp \langle \ln P \rangle_2 / Z_1 \qquad\qquad Q_2 = \exp \langle \ln P \rangle_1 / Z_2 \qquad (15)$$

If we substitute this form for $Q_2$ in (14) and use $Z_2 = \sum \exp \langle \ln P \rangle_1$, we obtain

$$\ln P(V) \geq - \langle \ln Q_1 \rangle_1 + \ln \sum_2 \exp \langle \ln P \rangle_1 \qquad (16)$$

In general, the final term may not have a simple form. In the case of approximating a BM $P$, $\ln P = s_1 \cdot J_{11} s_1 + 2 s_1 \cdot J_{12} s_2 + s_2 \cdot J_{22} s_2 - \ln Z_p$. Used in (16), we get:

$$\ln P(V) \geq - \langle \ln Q_1 \rangle_1 - \ln Z_p + \langle s_1 \cdot J_{11} s_1 \rangle_1 + \ln \sum_2 \exp \left( s_2 \cdot J_{22} s_2 + 2 s_2 \cdot J_{21} \langle s_1 \rangle_1 \right) \qquad (17)$$

so that the final term of (17) is the normalizing constant of a BM with connection matrix $J_{22}$ and whose diagonals are shifted by $J_{21} \langle s_1 \rangle_1$. One can therefore identify a set of nodes $s_1$ which, when eliminated, reveal a tractable structure on the nodes $s_2$. The nodes that were removed are compensated for by using a variational distribution $Q_1(s_1)$. If $P$ is a BM, then the optimal $Q_1$ has its weights fixed to those of $P$ restricted to variables $s_1$, but with variable biases shifted by $J_{12} \langle s_2 \rangle_2$. Restricting $Q_1$ to factorized models, we recover the node elimination bound [9] which can readily be improved by considering *non-factorized* distributions $Q_1$ (for example those introduced in this paper), see fig(3). Note, however, that there is no a-priori guarantee that using such partitioned approximations will lead to a better approximation than that obtained from a tractable variational distribution defined on the whole graph, but which does not have such a product form. Using a product of *conditional* distributions over clusters of nodes is developed more fully in [11].

### 6.2   Substructure Variation

The process of using a $Q$ defined on the whole graph but for which only a subset of the connections are adaptive is termed substructure variation [10]. In the context of BMs, Saul et al [2] identified weights in the original intractable distribution $P$ that, if set to zero, would lead to a tractable graph $Q(s) = P(s|h, J, J_{intractable} = 0)$. To compensate for these removed weights they allowed the biases in $Q$ to vary such that the KL divergence between $Q$ and $P$ is minimized. In general, this is a weaker method than one in which potentially all the parameters in the approximating network are adaptive, such as using a decimatable BM.

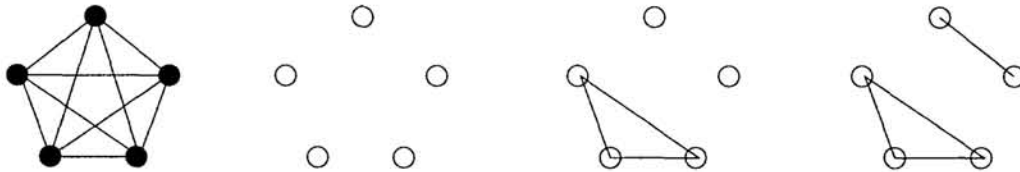

(a) Intractable Model (b) "Naive" mean field    (c) Node elimination    (d) Partioning

Figure 3: (a) A non-decimatable 5 node BM. (b) The standard factorized approximation. (c) Node Elimination (d) Partitioning, where a richer distribution is considered on the eliminated nodes. A solid line denotes a weight fixed to those in the original graph. A solid node is fixed, and an open node represents a variable bias.

## 7 Conclusion

Finding accurate, controllable approximations of graphical models is crucial if their application to large scale problems is to be realised. We have elucidated two general classes of tractable approximations, both based on the Kullback-Leibler divergence. Future interesting directions include extending the class of distributions to higher order Boltzmann Machines (for which the class of decimation rules is greater), and to mixtures of these approaches. Higher order perturbative approaches are considered in [12]. These techniques therefore facilitate the approximating power of tractable models which can lead to a considerable improvement in performance.

[1] E. Castillo, J. M. Gutierrez, and A. S. Hadi. *Expert Systems and Probabilistic Network Models*. Springer, 1997.

[2] L. K. Saul and M. I. Jordan. Boltzmann Chains and Hidden Markov Models. In G. Tesauro, D. S. Touretzky, and T. K. Leen, editors, Advances in Neural Information Processing Systems, pages 435–442. MIT Press, 1995. NIPS 7.

[3] T. Jaakkola. *Variational Methods for Inference and Estimation in Graphical Models*. PhD thesis, Massachusetts Institute of Technology, 1997.

[4] L. K. Saul, T. Jaakkola, and M. I. Jordan. Mean Field Theory for Sigmoid Belief Networks. *Journal of Artificial Intelligence Research*, 4:61–76, 1996.

[5] C.M. Bishop, N. Lawrence, T. Jaakkola, and M. I. Jordan. Approximating Posterior Distributions in Belief Networks using Mixtures. MIT Press, 1998. NIPS 10.

[6] C. Peterson and J. R. Anderson. A Mean Field Theory Learning Algorithm for Neural Networks. *Complex Systems*, 1:995–1019, 1987.

[7] Conrad C. Galland. The limitations of deterministic Boltzmann machine learning. *Network: Computation in Neural Systems*, 4:355–379, 1993.

[8] R. Neal. Connectionist learning of Belief Networks. *Artificial Intelligence*, 56:71–113, 1992.

[9] T. S. Jaakkola and M. I. Jordan. Recursive Algorithms for Approximating Probabilities in Graphical Models. MIT Press, 1996. NIPS 9.

[10] L. K. Saul and M. I. Jordan. Exploiting Tractable Substructures in Intractable Networks. MIT Press, 1996. NIPS 8.

[11] W. Wiegerinck and D. Barber. Mean Field Theory based on Belief Networks for Approximate Inference. 1998. ICANN 98.

[12] D. Barber and P. van de Laar. Variational Cumulant Expansions for Intractable Distributions. *Journal of Artificial Intelligence Research*, 1998. Accepted.

## Footnotes

*Real World Computing Partnership

†Foundation for Neural Networks

[1]In which $Q = \sum_j \lambda_j Q(H|\boldsymbol{\mu}^j)$ where $Q(H|\boldsymbol{\mu}) = \prod_i \mu_i^{H_i} (1 - \mu_i)^{1-H_i}$ and $\sum_j \lambda_j = 1$.

[2]In the case of fully connected BMs, for computing with a $Q$ which is the product of $K$ partitions (each of which is fully connected say), the computing time reduces from $2^N$ for the "intractable" $P$ to $K 2^{N/K}$ for $Q$, which can be a considerable reduction.
